# Efficient Relational Learning with Hidden Variable Detection

**Ni Lao, Jun Zhu, Liu Liu, Yandong Liu, William W. Cohen**
Carnegie Mellon University
5000 Forbes Avenue, Pittsburgh, PA 15213
{nlao,junzhu,liuliu,yandongl,wcohen}@cs.cmu.edu

## Abstract

Markov networks (MNs) can incorporate arbitrarily complex features in modeling relational data. However, this flexibility comes at a sharp price of training an exponentially complex model. To address this challenge, we propose a novel relational learning approach, which consists of a restricted class of relational MNs (RMNs) called relation tree-based RMN (treeRMN), and an efficient Hidden Variable Detection algorithm called Contrastive Variable Induction (CVI). On one hand, the restricted treeRMN only considers simple (e.g., unary and pairwise) features in relational data and thus achieves computational efficiency; and on the other hand, the CVI algorithm efficiently detects hidden variables which can capture long range dependencies. Therefore, the resultant approach is highly efficient yet does not sacrifice its expressive power. Empirical results on four real datasets show that the proposed relational learning method can achieve similar prediction quality as the state-of-the-art approaches, but is significantly more efficient in training; and the induced hidden variables are semantically meaningful and crucial to improve the training speed and prediction qualities of treeRMNs.

## 1 Introduction

Statistical relational learning has attracted ever-growing interest in the last decade, because of widely available relational data, which can be as complex as citation graphs, the World Wide Web, or relational databases. Relational Markov Networks (RMNs) are excellent tools to capture the statistical dependency among entities in a relational dataset, as has been shown in many tasks such as collective classification [22] and information extraction [18][2]. Unlike Bayesian networks, RMNs avoid the difficulty of defining a coherent generative model, thereby allowing tremendous flexibility in representing complex patterns [21]. For example, Markov Logic Networks [10] can be automatically instantiated as a RMN, given just a set of predicates representing attributes and relations among entities. The algorithm can be applied to tasks in different domains without any change. Relational Bayesian networks [22], in contrary, would require expert knowledge to design proper model structures and parameterizations whenever the schema of the domain under consideration is changed. However, this flexibility of RMN comes at a high price in training very complex models. For example, work by Kok and Domingos [10][11][12] has shown that a prominent problem of relational undirected models is how to handle the exponentially many features, each of which is an conjunction of several neighboring variables (or "ground atoms" in terms of first order logic). Much computation is spent on proposing and evaluating candidate features.

The main goal of this paper is to show that instead of learning a very expressive relational model, which can be extremely expensive, an alternative approach that explores Hidden Variable Detection (HVD) to compensate a family of restricted relational models (e.g., treeRMNs) can yield a very efficient yet competent relational learning framework. First, to achieve efficient inference, we introduce a restricted class of RMNs called relation tree-based RMNs (treeRMNs), which only considers unary (single variable assignment) and pairwise (conjunction of two variable assignments) features.

Since the Markov blanket of a variable is concisely defined by a relation tree on the schema, we can easily control the complexities of treeRMN models. Second, to compensate for the restricted expressive power of treeRMNs, we further introduce a hidden variable induction algorithm called Contrastive Variable Induction (CVI), which can effectively detect latent variables capturing long range dependencies. It has been shown in relational Bayesian networks [24] that hidden variables can help propagating information across network structures, thus reducing the burden of extensive structural learning. In this work, we explore the usefulness of hidden variables in learning RMNs. Our experiments on four real datasets show that the proposed relational learning framework can achieve similar prediction quality to the state-of-the-art RMN models, but is significantly more efficient in training. Furthermore, the induced hidden variables are semantically meaningful and are crucial to improving training speed of treeRMN.

In the remainder of this paper, we first briefly review related work and training undirected graphical models with mean field contrastive divergence. Then we present the treeRMN model and the CVI algorithm for variable induction. Finally, we present experimental results and conclude this paper.

## 2 Related Work

There has been a series of work by Kok and Domingos [10][11][12] developing Markov Logic Networks (MLNs) and showing their flexibility in different applications. The treeRMN model we introduced in this work is intended to be a simpler model than MLNs, which can be trained more efficiently, yet still be able to capture complex dependencies. Most of the existing RMN models construct Markov networks by applying templates to entity relation graphs [21][8]. The treeRMN model that we are going to introduce uses a type of template called a relation tree, which is very general and applicable to a wide range of applications. This relation tree template resembles the path-based feature generation approach for relational classifiers developed by Huang et al. [7].

Recently, much work has been done to induce hidden variables for generative Bayesian networks [5][4][16][9][20][14]. However, previous studies [6][19] have pointed out that the generality of Bayesian Networks is limited by their need for prior knowledge on the ordering of nodes. On the other hand, very little progress has been made in the direction of non-parametric hidden variable models based on discriminative Markov networks (MNs). One recent attempt is the Multiple Relational Clustering (MRC) [11] algorithm, which performs top-down clustering of predicates and symbols. However, it is computationally expensive because of its need for parameter estimation when evaluating candidate structures. The CVI algorithm introduced in this work is most similar to the "ideal parent" algorithm [16] for Gaussian Bayesian networks. The "ideal parent" evaluates candidate hidden variables based on the estimated gain of log-likelihood they can bring to the Bayesian network. Similarly, the CVI algorithm evaluates candidate hidden variables based on the estimated gain of an regularized RMN log-likelihood, thus avoids the costly step of parameter estimation.

## 3 Preliminaries

Before describing our model, let's briefly review undirected graphical models (a.k.a, Markov networks). Since our goal is to develop an efficient RMN model, we use the simple but very efficient mean field contrastive divergence [23] method. Our empirical results show that even the simplest naive mean field can yield very promising results. Extension to using more accurate (but also more expensive) inference methods, such as loopy BP [15] or structured mean fields can be done similarly.

Here we consider the general case that Markov networks have observed variables $\mathbf{O}$, labeled variables $\mathbf{Y}$, and hidden variables $\mathbf{H}$. Let $\mathbf{X} = (\mathbf{Y}, \mathbf{H})$ be the joint of hidden and labeled variables. The conditional distribution of $\mathbf{X}$ given observations $\mathbf{O}$ is $p(\mathbf{x}|\mathbf{o}; \theta) = \exp(\theta^\top \mathbf{f}(\mathbf{x}, \mathbf{o}))/Z(\theta)$, where $\mathbf{f}$ is a vector of feature functions $f_k$; $\theta$ is a vector of weights; $Z(\theta) = \sum_{\mathbf{x}} \exp(\theta^\top \mathbf{f}(\mathbf{x}, \mathbf{o}))$ is a normalization factor; and $f_k(\mathbf{x}, \mathbf{o})$ counts the number of times the $k$-th feature fires in $(\mathbf{x}, \mathbf{o})$. Here we assume that the range of each variable is discrete and finite. Many commonly used graphical models have tied parameters, which allow a small number of parameters to govern a large number of features. For example, in a linear chain CRF, each parameter is associated with a *feature template*: e.g. "the current node having label $y_t = 1$ and the immediate next neighbor having label $y_{t+1} = 1$". After applying each template to all the nodes in a graph, we get a graphical model with a large number of features (i.e., instantiations of feature templates). In general, a model's order of Markov dependence is determined by the maximal number of neighboring steps considered by any one of

its feature templates. In the context of relational learning, the templates can be defined similarly, except having richer representations–with multiple types of entities and neighboring relations.

Given a set of training samples $\mathcal{D} = \{(\mathbf{y}_m, \mathbf{o}_m)\}_{m=1}^M$, the parameter estimation of MN can be formulated as maximizing the following regularized log-likelihood

$$L(\theta) = \sum_{m=1}^M l_m(\theta) - \lambda\|\theta\|_1 - \frac{1}{2}\beta\|\theta\|_2^2, \tag{1}$$

where $\lambda$ and $\beta$ are non-negative regularization constants for the $\ell_1$ and $\ell_2$-norm respectively. Because of its singularity at the origin, the $\ell_1$-norm can yield a sparse estimate, which is a desired property for hidden variable discovery, as we shall see. The differentiable $\ell_2$-norm is useful when there are strongly correlated features. The composite $\ell_1/\ell_2$-norm is known as ElasticNet [27], which has been shown to have nice properties. The log-likelihood for a single sample is

$$l(\theta) = \log p(\mathbf{y}|\mathbf{o}; \theta) = \log \sum_{\mathbf{h}} p(\mathbf{h}, \mathbf{y}|\mathbf{o}; \theta), \tag{2}$$

and its gradient is $\nabla_\theta l(\theta) = \langle \mathbf{f} \rangle_{p_\mathbf{y}} - \langle \mathbf{f} \rangle_p$, where $\langle \cdot \rangle_p$ is the expectation under the distribution $p$. To simplify notation, we use $p$ to denote the distribution $p(\mathbf{h}, \mathbf{y}|\mathbf{o}; \theta)$ and $p_\mathbf{y}$ to denote $p(\mathbf{h}|\mathbf{y}, \mathbf{o}; \theta)$.

For simple (e.g. tree-structured) MNs, message passing algorithms can be used to infer the marginal probabilities as required in the gradients exactly. For general MNs, however, we need approximate strategies like variational or Monte Carlo methods. Here we use simple mean field variational method [23]. By analogy with statistical physics, the free energy of any distribution $q$ is defined as

$$F(q) = \langle -\theta^\top \mathbf{f} \rangle_q - H(q). \tag{3}$$

Therefore, $F(p) = -\log Z(\theta)$, $F(p_\mathbf{y}) = -\log \sum_{\mathbf{h}} \exp(\theta^\top \mathbf{f}(\mathbf{y}, \mathbf{h}, \mathbf{o}))$, and $l(\theta) = F(p) - F(p_\mathbf{y})$. Let $q_0$ be the mean field approximation of $p(\mathbf{h}, \mathbf{y}|\mathbf{o}; \theta)$ with $\mathbf{y}$ clamped to their true values, and $q_t$ be the approximation of $p(\mathbf{h}, \mathbf{y}|\mathbf{o}; \theta)$ obtained by applying $t$ steps of mean field updates to $q_0$ with $\mathbf{y}$ free. Then $F(q_0) \geq F(q_t) \geq F(q_\infty) \geq F(p)$. As in [23], we set $t = 1$, and use

$$l^{CD1}(\theta) \triangleq F(q_1) - F(q_0) \tag{4}$$

to approximate $l(\theta)$, and its gradient is $\nabla_\theta l^{CD1}(\theta) = \langle \mathbf{f} \rangle_{q_0} - \langle \mathbf{f} \rangle_{q_1}$. The new objective function $\mathcal{L}^{CD1}(\theta)$ uses $l^{CD1}(\theta)$ to replace $l(\theta)$. One advantage of CD is that it avoids $q$ being trapped in a possible multimodal distribution of $p(\mathbf{h}, \mathbf{y}|\mathbf{o}; \theta)$ [25][3]. With the above approximation, we can use orthant-wise L-BFGS [1] to estimate the parameters $\theta$.

## 4   Relation Tree-Based RMNs

In the following, we formally define the treeRMN model with relation tree templates, which is very general and applicable to a wide range of applications.

A *schema* $S$ (Figure 1 left) is a pair $(\mathbf{T}, \mathbf{R})$. $\mathbf{T} = \{T_i\}$ is a set of *entity types* which include both *basic entity types* (e.g., $Person$, $Class$) and *composite entity types* (e.g., $\langle Person, Person\rangle$, $\langle Person, Class\rangle$). Each entity type is associated with a set of *attributes* $A(T) = \{T.A_i\}$: e.g., $A(Person) = \{Person.gender\}$. $\mathbf{R} = \{R\}$ is a set of binary relations. We use $dom(R)$ to denote the domain type of $R$ and $range(R)$ to denote its range. For each argument of a composite entity type, we define two relations, one with outward direction (e.g. $PP1$ means from a Person-Person pair to its first argument) and another with inward direction (e.g. $PP1^{-1}$). Here we use $^{-1}$ to denote the inverse of a relation. We further introduce a $Twin$ relation, which connects a composite entity type to itself. Its semantics will be clear later. In principle, we can define other types of relations such as those corresponding to functions in second order logic (e.g. $Person \xrightarrow{FatherOf} Person$).

An *entity relation graph* $G = I_\mathbf{E}(S)$ (Figure 1 right), is the instantiation of schema $S$ on a set of basic entities $\mathbf{E} = \{e_i\}$. We define the *instantiation* of a basic entity type $T$ as $I_\mathbf{E}(T) = \{e : e.T = T\}$, and similarly for a composite type $I_\mathbf{E}(T = \langle T_1, ..., T_k\rangle) = \{\langle e_1, ..., e_k\rangle : e_i.T = T_i\}$. In the given example, $I_\mathbf{E}(Person) = \{p1, p2\}$ is the set of persons; $I_\mathbf{E}(Class) = \{c1\}$ is the set of classes; $I_\mathbf{E}(\langle Person, Person\rangle) = \{\langle p1, p2\rangle, \langle p2, p1\rangle\}$ is the set of person-person pairs; and $I_\mathbf{E}(\langle Person, Class\rangle) = \{\langle p1, c1\rangle, \langle p2, c1\rangle\}$ is the set of person-class pairs. Each entity $e$ has a set of variables $\{e.X_i\}$ that correspond to the set of attributes of its entity type $A(e.T)$. For a composite entity that consists of two entities of the same type, we'd like to capture its correlation with its *twin*– the composite entity made of the same basic entities but in reversed order. Therefore, we add the $Twin$ relation between all pairs of twin entities: e.g., from $\langle p1, p2\rangle$ to $\langle p2, p1\rangle$, and vice versa.

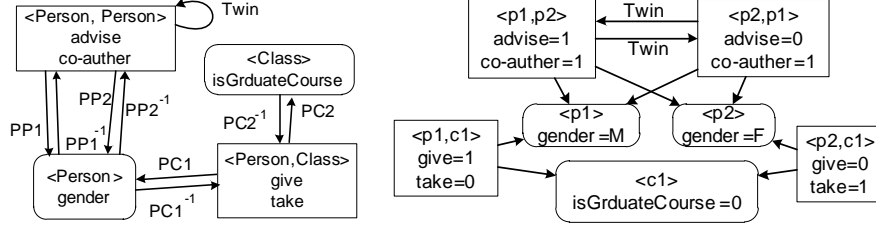

Figure 1: (Left) is a schema, where round and rectangular boxes represent basic and composite entity types respectively. (Right) is a corresponding entity relation graph with three basic entities: p1, p2, c1. For clarity we only show one direction of the relations and omit their labels.

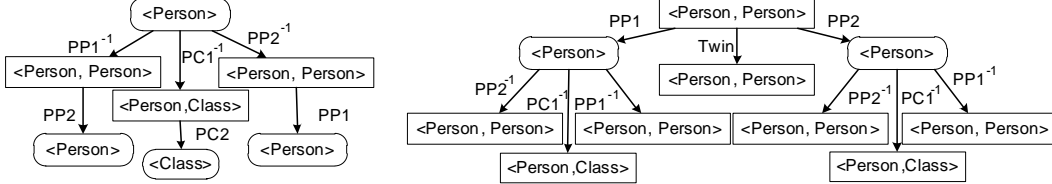

Figure 2: Two-level relation trees for the $Person$ type (left) and the $\langle Person, Person \rangle$ type (right).

Given a schema, we can conveniently express how one entity can reach another entity by the concept of a relation path. A *relation path* $P$ is a sequence of relations $R_1 \ldots R_\ell$ for which the domains and ranges of adjacent relations are compatible–i.e., $range(R_i) = dom(R_{i+1})$. We define $dom(R_1 \ldots R_\ell) \equiv dom(R_1)$ and $range(R_1 \ldots R_\ell) \equiv range(R_\ell)$, and when we wish to emphasize the types associated with each step in a path, we will write the path $P = R_1 \ldots R_\ell$ as $T_0 \xrightarrow{R_1} \ldots \xrightarrow{R_\ell} T_\ell$, where $T_0 = dom(R_1) = dom(P)$, $T_1 = range(R_1) = dom(R_2)$ and so on. Note that, because some of the relations reflect one-to-one mappings, there are groups of paths that are equivalent–e.g., the path $Person$ is actually equivalent to the path $Person \xrightarrow{PC1^{-1}} \langle Person, Class \rangle \xrightarrow{PC1} Person$. To avoid creating these uninteresting paths, we add a constraint to outward composite relations (e.g. $PP1, PC1$) that they cannot be immediately preceded by their inverse. We also constrain that the $Twin$ relation should not be combined with any other relations.

Now, the Markov blanket of an entity $e \in T$ can be concisely defined by the set of all relation paths with domain $T$ and of length $\leq \ell$ (as shown in Figure 2). We call this set the *relation tree* of type $T$, and denote it as $Tree(T, \ell) = \{P\}$. We define a *unary template* as $T.A_i = a$, where $A_i$ is an attribute of type $T$, and $a \in range(A_i)$. This template can be applied to any entity $e$ of type $T$ in the entity relation graph. We define a *pairwise template* as $T.A_i = a \bigwedge P.B_j = b$, where $A_i$ is an attribute of type $T$, $a \in range(A_i)$, $P.B_j$ is an attribute of type $range(P)$, $dom(P) = T$, and $b \in range(B_j)$. This template can be applied to any entity pair $(e_1, e_2)$, where $e_1.T = T$ and $e_2 \in e_1.P$. Here we define $e.P$ as the set of entities reachable from entity $e \in T$ through the relation path $P$. For example, the following template

$$pp.coauthor = 1 \bigwedge pp \xrightarrow{PP1} p \xrightarrow{PP1^{-1}} pp.advise = 1$$

can be applied to any person-person pair, and it fires whenever co-author=1 for this person pair, and the first person (identified as $pp \xrightarrow{PP1} p$ ) also have advise=1 with another person. Here we use $p$ as a shorthand for the type $Person$, and $pp$ a shorthand for $\langle Person, Person \rangle$. In our current implementation, we systematically enumerate all possible unary and pairwise templates.

Given the above concepts, we define a *treeRMN* model $\mathcal{M} = (G, \mathbf{f}, \theta)$ as the tuple of an entity relation graph $G$, a set of feature functions $\mathbf{f}$, and their weights $\theta$. Each feature function $f_k$ counts the number of times the $k$-th template fires in $G$. Generally, the complexity of inference is exponential in the depth of the relation trees, because both the number of templates and their sizes of Markov blankets grow exponentially w.r.t. the depth $\ell$. TreeRMN provides us a very convenient way to control the complexity by the single parameter $\ell$. Since treeRMN only considers pairwise and unary features, it is less expressive than Markov Logic Networks [10], which can define higher order features by conjunction of predicates; and treeRMN is also less expressive than relational Bayesian networks [9][20][14], which have factor functions with three arguments. However, the limited expressive power of treeRMN can be effectively compensated for by detecting hidden variables, which is another key component of our relational learning approach, as explained in the next section.

| **Algorithm 1** Contrastive Variable Induction | **Algorithm 2** Bottom Up Clustering of Entities |
|---|---|
| **initialize** a treeRMN $\mathcal{M} = (G, \mathbf{f}, \theta)$ <br> **while** true **do** <br> $\quad$ estimate parameters $\theta$ by L-BFGS <br> $\quad (\mathbf{f}', \theta') = induceHiddenVariables(\mathcal{M})$ <br> $\quad$ **if** no hidden variable is induced **then** <br> $\quad\quad$ **break** <br> $\quad$ **end if** <br> **end while** <br> **return** $\mathcal{M}$ | **initialize** clustering $\Gamma = \{I_i = \{i\}\}$ <br> **while** true **do** <br> $\quad$ **for** any pair of clusters $I_1, I_2 \in \Gamma$ **do** <br> $\quad\quad inc(I_1, I_2) = \Delta_{I_1 \cup I_2} - \Delta_{I_1} - \Delta_{I_2}$ <br> $\quad$ **end for** <br> $\quad$ **if** the largest increment $\leq 0$ **then** <br> $\quad\quad$ **break** <br> $\quad$ **end if** <br> $\quad$ merge the pair with the largest increment <br> **end while** <br> **return** $\Gamma$ |

## 5 Contrastive Variable Induction (CVI)

As we have explained in the previous section, in order to compensate for the limited expressive power of a shallow treeRMN and capture long-range dependencies in complex relational data, we propose to introduce hidden variables. These variables are detected effectively with the Contrastive Variable Induction (CVI) algorithm as explained below.

The basic procedure (Algorithm 1) starts with a treeRMN model on observed variables, which can be manually designed or automatically learned [13]; then it iteratively introduces new HVs to the model and estimate its parameters. The key to making this simple procedure highly efficient is a fast algorithm to evaluate and select good candidate HVs. We give closed-form expressions of the likelihood gain and the weights of newly added features under contrastive divergence approximation [23] (other type of inference can be done similarly). Therefore, the CVI process can be very efficient, only adding small overhead to the training of a regular treeRMN.

Consider introducing a new HV $H$ to the entity type $T$. In order for $H$ to influence the model, it needs to be connected to the existing model. This is done by defining additional feature templates: we can denote a HV candidate by a tuple $(\{q^{(i)}(H)\}, \mathbf{f}_H, \theta_H)$, where $\{q^{(i)}(H)\}$ is the set of distributions of the hidden variable $H$ on all entities of type $T$, $\mathbf{f}_H$ is a set of pairwise feature templates that connect $H$ to the existing model, and $\theta_H$ is a vector of feature weights. Here we assume that any feature $f \in \mathbf{f}_H$ is in the pairwise form $f_{H=1 \bigwedge A=a}$, where $a$ is the assignment to one of the existing variables $A$ in the relation tree of type $T$. Ideally, we would like to identify the candidate HV, which gives the maximal gain in the regularized objective function $\mathcal{L}^{CD1}(\theta)$.

For easy evaluation of $H$, we set its mean field variational parameters $\mu_H$ to either 0 or 1 on the entities of type $T$. This yields a lower bound to the gain of $\mathcal{L}^{CD1}(\theta)$. Therefore, a candidate HV can be represented as $(I, \mathbf{f}_H, \theta_H)$, where $I$ is the set of indices to the entities with $\mu_H = 1$. Using second order Taylor expansion, we can show that for a particular feature $f \in \mathbf{f}_H$ the maximal gain

$$\Delta_{I,f} = \frac{1}{2} \frac{\lfloor -e_I[f] \rfloor_\lambda^2}{\delta_I[f] + \beta} \tag{5}$$

is achieved at

$$\theta_f = \frac{\lfloor -e_I[f] \rfloor_\lambda}{\delta_I[f] + \beta}, \tag{6}$$

where $\lfloor\rfloor$ is a truncation operator: $\lfloor a \rfloor_b = a - b$, if $a > b$; $a + b$, if $a < -b$; 0, otherwise. Error $e_I[f] = \langle f \rangle_{q_1, I} - \langle f \rangle_{q_0, I}$ is the difference of $f$'s expectations, and $\delta_I[f] = Var^*_{q_1, I}[f] - Var^*_{q_0, I}[f]$ is the differences of $f$'s variances[1]. Here we use $q, I$ to denote the distribution $q$ of the existing variables augmented by the distribution of $H$ parameterized by the index set $I$. $q_0$ and $q_1$ are the wake and sleep distributions estimated by 1-step mean-field CD. The estimations in Eq. (5) and (6) are simple, yet have nice intuitive explanations about the effects of the $\ell_1$ and $\ell_2$ regularizer as used in Eq. (1): a large $\ell_2$-norm (i.e. large $\beta$) smoothly shrinks both the (estimated) likelihood gain and the feature weights; while the non-differentiable $\ell_1$-norm not only shrink the estimated gain and feature weights, but also drive features to have zero gains, therefore, can automatically select the features.

If we assume that the gains of individual features are independent, then the estimated gain for $H$ is

$$\Delta_I \approx \sum_{f \in \mathbf{f}_I} \Delta_{I,f},$$

where $\mathbf{f}_I = \{f : \Delta_{I,f} > 0\}$ is the set of features that are expected to improve the objective function. However, finding the index set $I$ that maximizes $\Delta_I$ is still non-trivial—an NP-hard combinatory optimization problem, which is often tackled by top-down or bottom-up procedures in the clustering literature. Algorithm 2 uses a simple bottom up clustering algorithm to build a hierarchy of clusters. It starts with each sample as an individual cluster, and then repeatedly merges the two clusters that lead to the best increment of gain. The merging is stopped if the best increment $\leq 0$.

After clustering, we introduce a *single* categorical variable that treats each cluster with positive gain as a category, and the remaining useless clusters are merged into a separate category. Introducing this categorical variable is equivalent to introducing a set of binary variables–one for each cluster with positive gain. From the above derivation, we can see that the essential part of the CVI algorithm is to compute the expectations and variances of RMN features, both of which can be done by any inference procedures, including the mean field as we have used. Therefore, in principle, the CVI algorithm can be extended to use other inference methods like belief propagation or exact inference.

**Remark 1** *after the induction step, the introduced HVs are treated as observations: i.e. their variational parameters are fixed to their initial 0 or 1 values. In the future, we'd like to treat the HVs as free variables. This can potentially correct the errors made by the greedy clustering procedure. The cardinalities of HVs may be adapted by operators like deleting, merging, or splitting of categories.*

**Remark 2** *currently, we only induce HVs to basic entity types. Extension to composite types can show interesting tenary relations such as "Abnormality can be PartOf Animals". However, this requires clustering over a much larger number of entities, which cannot be done by our simple implementation of bottom up clustering.*

## 6 Experiment

In this section, we present both qualitative and quantitative results of treeRMN model. We demonstrate that CVI can discover semantically meaningful hidden variables, which can significantly improve the speed and quality of treeRMN models.

### 6.1 Datasets

Table 1 shows the statistics of the four datasets used in our experiments. These datasets are commonly used by previous work in relational learning [9][11][20][14]. The **Animal** dataset contains a set of animals and their attributes. It consists exclusively of unary predicates of the form $A(a)$ where $A$ is an attribute and $a$ is an animal (e.g., Swims(Dolphin)). This is a simple propositional dataset with no relational structure, but is useful as a base case for comparison. The **Nation** dataset contains attributes of nations and relations among them. The binary predicates are of the form $R(n_1, n_2)$, where $n_1$, $n_2$ are nations and $R$ is a relation between them (e.g., *ExportsTo*, *GivesEconomicAidTo*). The unary predicates are of the form $A(n)$, where $n$ is a nation and $A$ is a attribute (e.g.,

| | Basic | | Composite | |
|---|---|---|---|---|
| | #E | #A | #E | #A |
| **Animal** | 50 | 80 | 0 | 0 |
| **Nation** | 14 | 111 | 196 | 56 |
| **UML** | 135 | 0 | 18,225 | 49 |
| **Kinship** | 104 | 0 | 10,816 | 1* |

Table 1: Number of entities (#E) and attributes (#A) for four datasets. *The kinship data has only one attribute which has 26 possible values.

$Communist(China)$). The **UML** dataset is a biomedical ontology called Unified Medical Language System. It consists of binary predicates of the form $R(c_1, c_2)$, where $c_1$ and $c_2$ are biomedical concepts and $R$ is a relation between them (e.g.,*Treats(Antibiotic,Disease)*). The **Kinship** dataset contains kinship relationships among members of the Alyawarra tribe from Central Australia. Predicates are of the form $R(p_1, p_2)$, where $R$ is a kinship term and $p_1, p_2$ are persons. Except for the animal data, the number of composite entities is the square of the number of basic entities.

### 6.2 Characterization of treeRMN and CVI

In this section, we analyze the properties of the discovered hidden variables and demonstrate the behavior of the CVI algorithm. For the simple non-relational Animal data, if we start with a full model with all pairwise features, CVI will decide not to introduce any hidden variables. If we run CVI starting from a model with only unary features, however, CVI decides to introduce one hidden variable $H_0$ with 8 categories. Table 2 shows the associated entities and features for the first four categories. We can see that they nicely identify marine mammals, predators, rodents, and primates.

| | Entities | Positive Features | Negative Features |
|---|---|---|---|
| **C0** | KillerWhale Seal Dolphin BlueWhale Walrus HumpbackWhale | Flippers Ocean Water Swims Fish Hairless Coastal Arctic ... | Quadrapedal Ground Furry Strainteeth Walks ... |
| **C1** | GrizzlyBear Tiger GermanShepherd Leopard Wolf Weasel Raccoon Fox Bobcat Lion | Stalker Fierce Meat Meatteeth Claws Hunter Nocturnal Paws Smart Pads ... | Timid Vegetation Weak Grazer Toughskin Hooves Domestic ... |
| **C2** | Hamster Skunk Mole Rabbit Rat Raccoon Mouse | Hibernate Buckteeth Weak Small Fields Nestspot Paws ... | Strong Muscle Big Toughskin ... |
| **C3** | SpiderMonkey Gorilla Chimpanzee | Tree Jungle Bipedal Hands Vegetation Forest ... | Plains Fields Patches ... |

Table 2: The associated entities and features (sorted by decreasing magnitude of feature weights) for the first four categories of the induced hidden variable $a.H_0$ on the Animal data. The features are in the form $a.H_0 = C_i \bigwedge a.A = 1$, where $A$ is any of the variables in the last two columns.

| | Entities | Positive Features |
|---|---|---|
| **C0** | AcquiredAbnormality AnatomicalAbnormality CongenitalAbnormality | $c \xrightarrow{CC2^{-1}} cc.\text{Causes} \quad c \xrightarrow{CC1^{-1}} cc.\text{PartOf}$ <br> $c \xrightarrow{CC2^{-1}} cc.\text{Complicates} \quad c \xrightarrow{CC2^{-1}} cc.\text{CooccursWith} \ldots$ |
| **C1** | Alga Plant | $c \xrightarrow{CC1^{-1}} cc.\text{InteractsWith} \quad c \xrightarrow{CC1^{-1}} cc.\text{LocationOf} \ldots$ |
| **C2** | Amphibian Animal Bird Invertebrate Fish Mammal Reptile Vertebrate | $c \xrightarrow{CC1^{-1}} cc.\text{InteractsWith} \quad c \xrightarrow{CC2^{-1}} cc.\text{PropertyOf}$ <br> $c \xrightarrow{CC2^{-1}} cc.\text{InteractsWith} \quad c \xrightarrow{CC2^{-1}} cc.\text{PartOf} \ldots$ |

Table 3: The associated entities and features (sorted by decreasing magnitude of feature weights) for the first three categories of the induced hidden variable $c.H_0$ on the UML data. The features are in the form $c.H_0 = C_i \bigwedge A = 1$, where $A$ is any of the variables in the last column.

For the three relational datasets, we use UML as an example. The induction process of Nation and Kinship datasets are similar, and we omit their details due to space limitation. For the UML task, CVI induces two multinomial hidden variables $H_0$ and $H_1$. As we can see from Figure 3, the inclusion of each hidden variable significantly improves the conditional log likelihood of the model. The first hidden variable $C.H_0$ has 43 categories, and Table 3 shows the top three of them. We can see that these categories represent the hidden concepts *Abnormalities*, *Animals* and *Plants* respectively. *Abnormalities* can be caused or treated by other concepts, and it can also be a part of other concepts. *Plants* can be the location of some other concepts; and some other concepts can be part of or the property of *Animals*. These grouping of concepts are similar to those reported by Kok and Domingos [11].

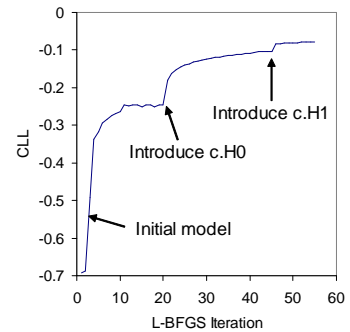

Figure 3: change of the conditional log likelihood during training for the UML data.

### 6.3 Overall Performance

Now we present quantitative evaluation of the treeRMN model, and compare it with other relational learning methods including MLN structure learning (MLS) [10], Infinite Relational Models (IRM) [9] and Multiple Relational Clustering (MRC) [11]. Following the methodology of [11], we situate our experiment in prediction tasks. We perform 10 fold cross validation by randomly splitting all the variables into 10 sets. At each run, we treat one fold as hidden during training, and then evaluate the prediction of these variables conditioned on the observed variables during testing. The overall performance is measured by training time, average Conditional Log-Likelihood (CLL), and Area Under the precision-recall Curve (AUC) [11]. All implementation is done with Java 6.0.

Table 4 compares the overall performance of treeRMN (RMN), treeRMN with hidden variable discovery (RMN$^{CVI}$), and other relational models (MSL, IRM and MRC) as reported in [11]. We use subscripts (0, 1, 2) to indicate the order of Markov dependency (depth of relation trees), and $\dim_\theta$ for the number of parameters. First, we can see that, without HVs, the treeRMNs with higher Markov orders generally perform better in terms of CLL and AUC. However, due to the complexity of high-order treeRMNs, this comes with large increases in training time. In some cases (e.g., Kinship data), a high order treeRMN can perform worse than a low order treeRMN probably due to the difficulty of inference with a large number of features. Second, training a treeRMN with CVI

| | Animal, $\lambda=0.01$, $\beta=1$ | | | | | Nation, $\lambda=0.01$, $\beta=1$ | | | |
|---|---|---|---|---|---|---|---|---|---|
| | CLL | AUC | $\dim_\theta$ | Time | | CLL | AUC | $\dim_\theta$ | Time |
| **RMN$_0$** | -0.34±0.03 | 0.88±0.02 | 3,655 | 5s | **RMN$_0$** | -0.40±0.01 | 0.63±0.04 | 7,812 | 15s |
| | | | | | **RMN$_1$** | -0.33±0.02 | 0.72±0.04 | 21,840 | 70s |
| | | | | | **RMN$_2$** | -0.38±0.03 | 0.71±0.04 | 40,489 | 446s |
| **RMN$_0^{CVI\star}$** | **-0.33±0.02** | **0.89±0.02** | 4,349 | 9s | **RMN$_1^{CVI}$** | **-0.31±0.02** | **0.83±0.04** | 22,191 | 104s |
| **MSL** | -0.54±0.04 | 0.68±0.04 | | †24h | **MSL** | -0.33±0.04 | 0.77±0.04 | | †24h |
| **MRC** | -0.43±0.04 | 0.80±0.04 | | †10h | **MRC** | **-0.31±0.02** | 0.75±0.03 | | †10h |
| **IRM** | -0.43±0.06 | 0.79±0.08 | | †10h | **IRM** | -0.32±0.02 | 0.75±0.03 | | †10h |
| | UML, $\lambda=0.01$, $\beta=10$ | | | | | Kinship, $\lambda=0.01$, $\beta=10$ | | | |
| | CLL | AUC | $\dim_\theta$ | Time | | CLL | AUC | $\dim_\theta$ | Time |
| **RMN$_0$** | -0.056±0.005 | 0.70±0.02 | 1,081 | 0.3h | **RMN$_0$** | §-2.95±0.01 | 0.08±0.00 | 25 | 6s |
| **RMN$_1$** | -0.044±0.002 | 0.68±0.04 | 2,162 | 1.0h | **RMN$_1$** | §-1.36±0.05 | 0.66±0.03 | 350 | 107s |
| **RMN$_2$** | -0.028±0.003 | 0.71±0.02 | 6,440 | 14.5h | **RMN$_2$** | §-2.34±0.01 | 0.33±0.00 | 1,625 | 2.1h |
| **RMN$_1^{CVI\star}$** | -0.005±0.001 | 0.94±0.01 | 6,946 | 453s | **RMN$_1^{CVI}$** | §-1.04±0.03 | 0.81±0.01 | 900 | 402s |
| **MSL** | -0.025±0.002 | 0.47±0.06 | | †24h | **MSL** | -0.066±0.006 | 0.59±0.08 | | †24h |
| **MRC** | **-0.004±0.000** | **0.97±0.00** | | †10h | **MRC** | **-0.048±0.002** | **0.84±0.01** | | †10h |
| **IRM** | -0.011±0.001 | 0.79±0.01 | | †10h | **IRM** | -0.063±0.002 | 0.68±0.01 | | †10h |

Table 4: Overall performance. Bold identifies the best performance, and ± marks the standard deviations. Experiments are conducted with Intel Xeon 2.33GHz CPU (E5410). ⋆These results were started with a treeRMN that only has unary features. §The CLL of kinship data is not comparable to previous approaches, because we treat each of its labels as one variable with 26 categories instead of 26 binary variables. †The results of existing methods were run on different machines (Intel Xeon 2.8GHz CPU), and their 10-fold data splits are independent to those used for the RMN models. They were allowed to run up to 10-24 hours, and here we assumes that these methods cannot achieve similar accuracy when the amount of training time is significantly reduced.

is only 2∼4 times slower than training a treeRMN of the same order of Markov dependency. On all three relational datasets, treeRMNs with CVI can significantly improve CLL and AUC. For the simple Animal dataset, the improvement is less significant because there is no long range dependency to be captured in this data. Although the CVI models have similar number features as the second order treeRMNs, their inferences are much faster due to their much smaller Markov blankets. Finally, on all datasets, the treeRMNs with CVI can achieve similar prediction quality as the existing methods (i.e., MSL, IRM and MRC), but is about two orders of magnitude more efficient in training. Specifically, it achieves significant improvements on the Animal and Nation data, but moderately worse results on the UML and Kinship data. Since both UML and Kinship data have no attributes in basic entity types, composite entities become more important to model. Therefore, we suspect that the MRC model achieves better performance because it can perform clustering on two-argument predicates which corresponds to composite entities.

# 7 Conclusions and Future Work

We have presented a novel approach for efficient relational learning, which consists of a restricted class of Relational Markov Networks (RMN) called relation tree-based RMN (treeRMN) and an efficient hidden variable induction algorithm called Contrastive Variable Induction (CVI). By using simple treeRMNs, we achieve computational efficiency, and CVI can effectively detect hidden variables, which compensates for the limited expressive power of treeRMNs. Experiments on four real datasets show that the proposed relational learning approach can achieve state-of-the-art prediction accuracy and is much faster than existing relational Markov network models.

We can improve the presented approach in several aspects. First, to further speedup the treeRMN model we can apply efficient Markov network feature selection methods [17][26] instead of systematically enumerating all possible feature templates. Second, as we have explained at the end of section 5, we'd like to apply HVD on composite entity types. Third, we'd also like to treat the introduced hidden variables as free variables and to make their cardinalities adaptive. Finally, we would like to explore high order features which involves more than two variable assignments.

**Acknowledgements.**
We gratefully acknowledge the support of NSF grant IIS-0811562 and NIH grant R01GM081293.

## Footnotes

[1] $Var_{q,I}[f]$ is intractable when we have tied parameters. Therefore, we approximate it by assuming that the occurrences of $f$ are independent to each other: i.e. $Var^*_{q,I}[f] = \sum_{V \in \mathbf{V}} Var_{q,I}[f(V)] = \sum_{V \in \mathbf{V}} \langle f(V) \rangle_{q,I} (1 - \langle f(V) \rangle_{q,I})$, where $V$ is any specific subset of variables that $f$ can be applied to.

# References

[1] Galen Andrew and Jianfeng Gao. Scalable training of $\ell_1$-regularized log-linear models. In *ICML*, 2007.

[2] Razvan C. Bunescu and Raymond J. Mooney. Collective information extraction with relational Markov networks. In *ACL*, 2004.

[3] Miguel A. Carreira-Perpinan and Geoffrey E. Hinton. On contrastive divergence learning. In *AISTATS*, 2005.

[4] Gal Elidan and Nir Friedman. The information bottleneck em algorithm. In *UAI*, 2003.

[5] Gal Elidan, Noam Lotner, Nir Friedman, and Daphne Koller. Discovering hidden variables: A structure-based approach. In *NIPS*, 2000.

[6] Nir Friedman, Lise Getoor, Daphne Koller, and Avi Pfeffer. Learning probabilistic relational models. In *IJCAI*, 1999.

[7] Yi Huang, Volker Tresp, and Stefan Hagen Weber. Predictive modeling using features derived from paths in relational graphs. In *Technical report*, 2007.

[8] Ariel Jaimovich, Ofer Meshi, and Nir Friedman. Template-based inference in symmetric relational Markov random fields. In *UAI*, 2007.

[9] Charles Kemp, Joshua B. Tenenbaum, Thomas L. Griffiths, Takeshi Yamada, and Naonori Ueda. Learning systems of concepts with an infinite relational model. In *AAAI*, 2006.

[10] Stanley Kok and Pedro Domingos. Learning the structure of Markov logic networks. In *ICML*, 2005.

[11] Stanley Kok and Pedro Domingos. Statistical predicate invention. In *ICML*, 2007.

[12] Stanley Kok and Pedro Domingos. Learning Markov logic networks using structural motifs. In *ICML*, 2010.

[13] Su-In Lee, Varun Ganapathi, and Daphne Koller. Efficient structure learning of Markov networks using $\ell_1$-regularization. In *NIPS*, 2006.

[14] Kurt T. Miller, Thomas L. Griffiths, and Michael I. Jordan. Nonparametric latent feature models for link prediction. In *NIPS*, 2009.

[15] Kevin P. Murphy, Yair Weiss, and Michael I. Jordan. Loopy belief propagation for approximate inference: An empirical study. In *UAI*, 1999.

[16] Iftach Nachman, Gal Elidan, and Nir Friedman. "Ideal parent" structure learning for continuous variable networks. In *UAI*, 2004.

[17] Simon Perkins, Kevin Lacker, and James Theiler. Grafting: Fast, incremental feature selection by gradient descent in function spaces. In *JMLR*, 2003.

[18] Hoifung Poon and Pedro Domingos. Joint inference in information extraction. In *AAAI*, 2007.

[19] Karen Sachs, Omar Perez, Dana Peer, Douglas A. Lauffenburger, and Garry P. Nolan. Causal protein-signaling networks derived from multiparameter single-cell data. In *Science*, 2005.

[20] Ilya Sutskever, Ruslan Salakhutdinov, and Josh Tenenbaum. Modelling relational data using Bayesian clustered tensor factorization. In *NIPS*, 2009.

[21] Benjamin Taskar, Pieter Abbeel, and Daphne Koller. Discriminative probabilistic models for relational data. In *UAI*, 2002.

[22] Benjamin Taskar, Eran Segal, and Daphne Koller. Probabilistic classification and clustering in relational data. In *IJCAI*, 2001.

[23] Max Welling and Geoffrey E. Hinton. A new learning algorithm for mean field Boltzmann machines. In *ICANN*, 2001.

[24] Zhao Xu, Volker Tresp, Kai Yu, and Hans-Peter Kriegel. Infinite hidden relational models. In *UAI*, 2006.

[25] Alan Yuille. The convergence of contrastive divergence. In *NIPS*, 2004.

[26] Jun Zhu, Ni Lao, and Eric P. Xing. Grafting-light: Fast, incremental feature selection and structure learning of Markov random fields. In *KDD*, 2010.

[27] Hui Zou and Trevor Hastie. Regularization and variable selection via the elastic net. In *Journal Of The Royal Statistical Society Series B*, 2005.

